# Multiple Threshold Neural Logic

**Vasken Bohossian**          **Jehoshua Bruck**

California Institute of Technology
Mail Code 136-93
Pasadena, CA 91125
E-mail: {vincent, bruck}@paradise.caltech.edu

## Abstract

We introduce a new Boolean computing element related to the Linear Threshold element, which is the Boolean version of the neuron. Instead of the sign function, it computes an arbitrary (with polynomialy many transitions) Boolean function of the weighted sum of its inputs. We call the new computing element an $LTM$ element, which stands for Linear Threshold with Multiple transitions.

The paper consists of the following main contributions related to our study of $LTM$ circuits: (i) the creation of efficient designs of $LTM$ circuits for the addition of a multiple number of integers and the product of two integers. In particular, we show how to compute the addition of $m$ integers with a single layer of $LTM$ elements. (ii) a proof that the area of the VLSI layout is reduced from $O(n^2)$ in $LT$ circuits to $O(n)$ in $LTM$ circuits, for $n$ inputs symmetric Boolean functions, and (iii) the characterization of the computing power of $LTM$ relative to $LT$ circuits.

## 1  Introduction

Human brains are by far superior to computers in solving hard problems like combinatorial optimization and image and speech recognition, although their basic building blocks are several orders of magnitude slower. This observation has boosted interest in the field of artificial neural networks [Hopfield 82], [Rumelhart 82]. The latter are built by interconnecting artificial neurons whose behavior is inspired by that of biological neurons. In this paper we consider the Boolean version of an artificial neuron, namely, a Linear Threshold ($LT$) element, which computes a neural-like

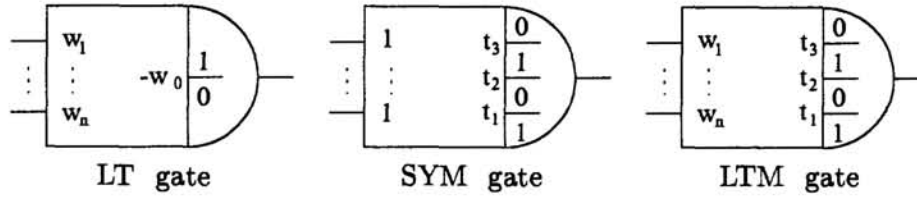

Figure 1: Schematic representation of *LT*, *SYM* and *LTM* computing elements.

Boolean function of $n$ binary inputs [Muroga 71]. An *LT* element outputs the sign of a weighted sum of its Boolean inputs. The main issues in the study of networks (circuits) consisting of *LT* elements, called *LT* circuits, include the estimation of their computational capabilities and limitations and the comparison of their properties with those of traditional Boolean logic circuits based on AND, OR and NOT gates (called *AON* circuits). For example, there is a strong evidence that *LT* circuits are more efficient than *AON* circuits in implementing a number of important functions including the addition, product and division of integers [Siu 94], [Siu 93].

Motivated by our recent work on the VLSI implementation of LT elements [Bohossian 95b], we introduce in this paper a more powerful computing element, a multiple threshold neuron, which we call *LTM*, which stands for Linear Threshold with Multiple transitions, see [Haring 66] and [Olafsson 88]. Instead of the sign function in the LT element it computes an arbitrary (with polynomialy many transitions) Boolean function of the weighted sum of its inputs.

The main issues in the study of *LTM* circuits (circuits consisting of *LTM* elements) include the estimation of their computational capabilities and limitations and the comparison of their properties to those of *AON* circuits. A natural approach in this study is first to understand the relation between *LT* circuits and *LTM* circuits. Our main contributions in this paper are:

- We demonstrate the power of *LTM* by deriving efficient designs of *LTM* circuits for the addition of $m$ integers and the product of two integers.

- We show that *LTM* circuits are more amenable in implementation than *LT* circuits. In particular, the area of the VLSI layout is reduced from $O(n^2)$ in *LT* circuits to $O(n)$ in *LTM* circuits, for $n$ input symmetric Boolean functions.

- We characterize the computing power of *LTM* relative to *LT* circuits.

Next we describe the formal definitions of *LT* and *LTM* elements.

## 1.1 Definitions and Examples

**Definition 1** *(Linear Threshold Gate – LT)*
*A linear threshold gate computes a Boolean function of its binary inputs :*

$$f(X) = sgn(w_0 + \sum_{i=1}^{n} w_i x_i)$$

*where the $w_i$ are integers and $sgn(.)$ outputs 1 if its argument is greater or equal to 0, and 0 otherwise.*

Figure 1 shows a $n$-input $LT$ element; if $\sum_1^n w_i x_i \geq -w_0$ the element outputs 1, otherwise it outputs 0. A single $LT$ gate is unable to compute parity. The latter belongs to the general class of symmetric functions – $SYM$.

**Definition 2** *(Symmetric Functions – $SYM$)*
*A Boolean function $f$ is symmetric if its value depends only on the number of ones in the input denoted by $|X|$.*

Figure 1 shows an example of a symmetric function; it has three transitions, it outputs 1 for $|X| < t_1$ and for $t_2 \leq |X| < t_3$, and 0 otherwise. *AND*, *OR* and parity are examples of symmetric functions. A single $LT$ element can implement only a limited subset of symmetric functions. We define $LTM$ as a generalization of $SYM$. That is, we allow the weights to be arbitrary as in the case of $LT$, rather than fixed to 1 (see Figure 1 ).

**Definition 3** *(Linear Threshold Gate with Multiple Transitions – $LTM$)*
*A function $f$ is in $LTM$ if there exists a set of weights $w_i \in Z$, $1 \leq i \leq n$ and a function $h : Z \longrightarrow \{0,1\}$ such that*

$$f(X) = h(\sum_{i=1}^{n} w_i x_i) \text{ for all } X \in \{0,1\}^n$$

*The only constraint on $h$ is that it undergoes polynomialy many transitions as its input scans $[-\sum_{i=1}^{n} |w_i|, \sum_{i=1}^{n} |w_i|]$.*

Notice that without the constraint on the number of transitions, an $LTM$ gate is capable of computing any Boolean function. Indeed, given an arbitrary function $f$, let $w_i = 2^{i-1}$ and $h(\sum_1^n 2^{i-1} x_i) = f(x_1, ..., x_n)$.

**Example 1** *($XOR \in LTM$)*
$XOR(X)$ outputs 1 if $|X|$, the number of 1's in $X$, is odd. Otherwise it outputs 0. To implement it choose $w_i = 1$ and $h(k) = \frac{1}{2}(1 - (-1)^k)$ for $0 \leq k \leq n$. Note that $h(k)$ needs not be defined for $k < 0$ and $k > n$, and has polynomialy many transitions.

Another useful function that $LTM$ can compute is $ADD(X, Y)$, the sum of two $n$-bit integers $X$ and $Y$.

**Example 2** *($ADD \in LTM$)*
To implement addition we set $f_l(X, Y) = h_l(\sum_{i=1}^{l} 2^i (x_i + y_i))$ where $h_l(k) = 1$ for $k \in [2^l, 2 \times 2^l - 1] \cup [3 \times 2^l, +\infty)$. Defined thus, $f_l$ computes the $m$-th bit of $X + Y$.

## 1.2  Organization

The paper is organized as follows. In Section 2, we study a number of applications as well as the VLSI implementations of $LTM$ circuits. In particular, we show how to compute the addition of $m$ integers with a single layer of $LTM$ elements. In Section 3, we prove the characterization results of $LTM$ – inclusion relations, in particular $LTM \subseteq \widehat{LT_2}$. In addition, we indicate which inclusions are proper and exhibit functions to demonstrate the separations.

## 2 *LTM* Constructions

The theoretical results about *LTM* can be applied to the VLSI implementation of Boolean functions. The idea of a gate with multiple thresholds came to us as we were looking for an efficient VLSI implementation of symmetric Boolean functions. Even though a single *LT* gate is not powerful enough to implement any symmetric function, a 2-layer *LT* circuit is. Furthermore, it is well known that such a circuit performs much better than the traditional logic circuit based on *AND*, *OR* and *NOT* gates. The latter has exponential size (or unbounded depth) [Wegener 91].

**Proposition 4** ( *LT₂ versus LTM for symmetric function implementation* )
*The $LT_2$ layout of a symmetric function requires area of $O(n^2)$, while using LTM one needs only area of $O(n)$.*

PROOF:
Implementing a generalized symmetric function in $LT_2$ requires up to $n$ *LT* gates in the first layer. Those have the same weights $w_i$ except for the threshold $w_0$. Instead of laying out $n$ times the same linear sum $\sum_1^n w_i x_i$ we do it once and compare the result to $n$ different thresholds. The resulting circuit corresponds to a single *LTM* gate. □

The $LT_2$ layout is redundant, it has $n$ copies of each weight, requiring area of at least $O(n^2)$. On the other hand, *LTM* performs a single weighted sum, its area requirement is $O(n)$.

A single *LTM* gate can compute the addition of $m$ $n$-bit integers *MADD*. The only constraint is that $m$ be polynomial in $n$.

**Theorem 5** *(MADD ∈ LTM)*
*A single layer of LTM gates can compute the sum of $m$ $n$-bit integers, provided that $m$ is at most polynomial in $n$.*

PROOF:
*MADD* returns an integer of at most $n + \log m$ bits. We need one *LTM* gate per bit. The least significant bit is computed by a simple $m$-bit *XOR*. For all other bits we use $f_l(X^{(1)}, .., X^{(m)}) = h_l(\sum_{i=1}^{l} 2^i \sum_{j=1}^{m} x_i^{(j)})$ to compute the $l$-th bit of the sum. □

**Corollary 6** *(PRODUCT ∈ PTM) A single layer of PTM ( which is defined below) gates, can compute the product of $m$ $n$-bit integers, provided that $m$ is at most polynomial in $n$.*

PROOF:
By analogy with $PT_1$, defined in [Bruck 90], in $PTM_1$ (or simply $PTM$) we allow a polynomial rather than a linear sum : $f(X) = h(w_1 x_1 + ... + w_n x_n + w_{(1,2)} x_1 x_2 + ...)$ However we restrict the sum to have polynomialy many terms (else, any Boolean function could be realized with a single gate). The product of two $n$-bit integers $X$ and $Y$ can be written as $PRODUCT(X, Y) = \sum_{i=1}^{n} x_i Y$. We use the construction of *MADD* in order to implement *PRODUCT*. $PRODUCT(X, Y) = MADD(x_1 Y, x_2 Y, ..., x_n Y)$. $f_l(X, Y) = h_l(\sum_{j=1}^{n} \sum_{i=1}^{l} 2^i x_j y_i)$ $f_l$ outputs the $l$-th bit of the product. □

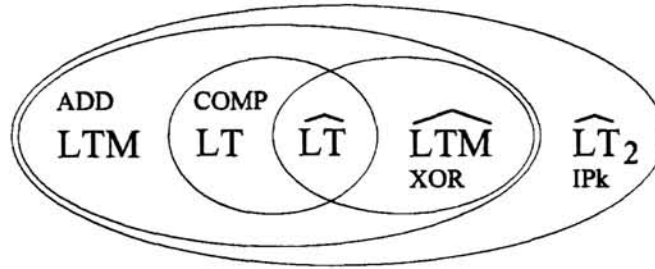

Figure 2: Relationship between Classes

## 3    Classification of $LTM$

We use a hat to indicate small (polynomialy growing) weights, e.g. $\widehat{LT}$, $\widehat{LTM}$ [Bohossian 95a], [Siu 91], and a subscript to indicate the depth (number of layers) of the circuit of more than a single layer. All the circuits we consider in this paper are of polynomial size (number of elements) in $n$ (number of inputs). For example, the class $\widehat{LT}_2$ consists of those Boolean functions that can be implemented by a depth-2 polynomial size circuit of $\widehat{LT}$ elements.

Figure 2 depicts the membership relations between five classes of Boolean functions, including, $LT$, $\widehat{LT}$, $LTM$, $\widehat{LTM}$ and $\widehat{LT}_2$, along with the functions used to establish the separations.

In this section we will prove the relations illustrated by Figure 2 .

**Theorem 7** ( *Classification of LTM* )
*The inclusions and separations shown in Figure 2 hold. That is,*

1. $\widehat{LT} \subseteq LT \subseteq LTM$

2. $\widehat{LT} \subseteq \widehat{LTM} \subseteq LTM$

3. $LTM \subseteq \widehat{LT}_2$

4. $XOR \in \widehat{LTM}$ *but* $XOR \notin LT$

5. $COMP \in LT$ *but* $COMP \notin \widehat{LTM}$

6. $ADD \in LTM$ *but* $ADD \notin LT \cup \widehat{LTM}$

7. $IP_k \in \widehat{LT}_2$ *but* $IP_k \notin LTM$

PROOF:
We show only the outline of the proof. The complete version can be found in [Bohossian 96]. Claims 1 and 2 follow from the definition. The first part of Claim 4 was shown in Example 1 and the second is well known. In Claim 5, $COMP$ stands for the Comparison function, the proof uses the pigeonhole principle and is related to the proof of $COMP \notin \widehat{LT}$ which can be found in [Siu 91]. In Claim 6 to show that $ADD \notin \widehat{LTM}$ we use the same idea as for $COMP$. Claim 3 is proved using a result from [Goldman 93]: a single $LT$ gate with arbitrary weights can be realized by an $\widehat{LT}_2$ circuit. Claim 7 introduces the function $IP_k(X,Y) = 1$ iff $\sum_1^n x_i y_i \geq k$,

0 otherwise. If $IP_k \in LTM$, using the result from [Goldman 93], we can construct a $\widehat{LT}_2$ circuit that computes $IP2$ (Inner Product mod 2) which is known to be false [Hajnal 94]. $\square$

What remains to be shown in order to complete the classification picture is $\widehat{LT} = LT \cap \widehat{LTM}$. We conjecture that this is true.

## 4 Conclusions

Our original goal was to use theoretical results in order to efficiently lay out a generalized symmetric function. During that process we came to the conclusion that the $LT_2$ implementation is partially redundant, which lead to the definition of $LTM$, a new, more powerful computing element. We characterized the power of $LTM$ relative to $LT$. We showed how it can be used to reduce the area of VLSI layouts from $O(n^2)$ to $O(n)$ and derive efficient designs for multiple addition and product. Interesting directions for future investigation are (i) to prove the conjecture : $\widehat{LT} = LT \cap \widehat{LTM}$, (ii) to apply spectral techniques ([Bruck 90]) to the analysis of $LTM$, in particular show how $PTM$ fits into the classification picture (Figure 2 ).

Another direction for future research consists in introducing the ideas described above in the domain of VLSI. We have fabricated a programmable generalized symmetric function on a $2\mu$, analog chip using the model described above. Floating gate technology is used to program the weights. We store a weight on a single transistor by injecting and tunneling electrons on the floating gate [Hasler 95].

## Acknowledgments

This work was supported in part by the NSF Young Investigator Award CCR-9457811 and by the Sloan Research Fellowship.

## References

[Bohossian 95a] V. Bohossian and J. Bruck. On Neural Networks with Minimal Weights. In *Advances in Neural Information Processing Systems 8*, MIT Press, Cambridge, MA, 1996, pp.246-252.

[Bohossian 95b] V. Bohossian, P. Hasler and J. Bruck. Programmable Neural Logic. *Proceedings of the second annual IEEE International Conference on Innovative Systems in Silicon*, pp. 13-21, October 1997.

[Bohossian 96] V. Bohossian and J. Bruck. Multiple Threshold Neural Logic. *Technical Report*, ETR010, June 1996. (available at http://paradise.caltech.edu/ETR.html)

[Bruck 90] J. Bruck. Harmonic Analysis of Polynomial Threshold Functions. *SIAM J. Disc. Math*, Vol. 3(No. 2)pp. 168–177, May 1990.

[Goldman 93] M. Goldmann and M. Karpinski. Simulating threshold circuits by majority circuits. In *Proc. 25th ACM STOC*, pages pp. 551–560, 1993.

[Hajnal 94] A. Hajnal, W. Maass, P. Pudlak, M. Szegedy, G. Turan. Threshold Circuits of Bounded Depth. *Journal of Computer and System Sciences*, Vol. 46(No. 2):pp. 129–154, April 1993.

[Haring 66] D.R. Haring. Multi-Threshold Threshold Elements. *IEEE Transactions on Electronic Computers*, Vol. EC-15, No. 1, February 1966.

[Hasler 95] P. Hasler, C. Diorio, B.A. Minch and C.A. Mead. Single Transistor Learning Synapses. *Advances in Neural Information Processing Systems 7*, MIT Press, Cambridge, MA, 1995, pp.817-824.

[Hastad 94] J. Hastad. On the size of weights for threshold gates. *SIAM. J. Disc. Math.*, 7:484–492, 1994.

[Hofmeister 96] T. Hofmeister. A Note on the Simulation of Exponential Threshold Weights. *1996 CONCOON conference.*

[Hopfield 82] J. Hopfield. Neural networks and physical systems with emergent collective computational abilities. *Proc. of the USA National Academy of Sciences*, 79:2554–2558, 1982.

[Muroga 71] M. Muroga. *Threshold Logic and its Applications*. Wiley-Interscience, 1971.

[Olafsson 88] S. Olafsson and Y.S. Abu-Mostafa. The Capacity of Multilevel Threshold Functions. *IEEE Transactions on Pattern Analysis and Machine Intelligence*, Vol.10, No. 2, March 1988.

[Rumelhart 82] D. Rumelhart and J. McClelland. Parallel distributed processing: Explorations in the microstructure of cognition. *MIT Press*, 1982.

[Siu 91] K. Siu and J. Bruck. On the power of threshold circuits with small weights. *SIAM J. Disc. Math.*, Vol. 4(No. 3):pp. 423–435, August 1991.

[Siu 93] K. Siu, J. Bruck, T. Kailath, and T. Hofmeister. Depth Efficient Neural Networks for Division and Related Problems. *IEEE Trans. on Information Theory*, Vol. 39(No. 3), May 1993.

[Siu 94] K. Siu and V.P. Roychowdhury. On Optimal Depth Threshold Circuits for Multiplication and Related Problems. *SIAM J. Disc. Math.*, Vol. 7(No. 2):pp. 284–292, May 94.

[Szegedy 89] M. Szegedy. Algebraic Methods in Lower Bounds for Computational Models with Limited Communication. *PhD Thesis*, Dep. Computer Science, Chicago Univ., December 1989.

[Wegener 91] I. Wegener. The complexity of the parity function in unbounded fan-in unbounded depth circuits. In *Theoretical Computer Science*, Vol. 85, pp. 155–170, 1991.
